# Oblivious Equilibrium: A Mean Field Approximation for Large-Scale Dynamic Games

**Gabriel Y. Weintraub, Lanier Benkard, and Benjamin Van Roy**
Stanford University
`{gweintra,lanierb,bvr}@stanford.edu`

## Abstract

We propose a mean-field approximation that dramatically reduces the computational complexity of solving stochastic dynamic games. We provide conditions that guarantee our method approximates an equilibrium as the number of agents grow. We then derive a performance bound to assess how well the approximation performs for any given number of agents. We apply our method to an important class of problems in applied microeconomics. We show with numerical experiments that we are able to greatly expand the set of economic problems that can be analyzed computationally.

## 1 Introduction

In this paper we consider a class of infinite horizon non-zero sum stochastic dynamic games. At each period of time, each agent has a given state and can make a decision. These decisions together with random shocks determine the evolution of the agents' states. Additionally, agents receive profits depending on the current states and decisions. There is a literature on such models which focusses on computation of Markov perfect equilibria (MPE) using dynamic programming algorithms. A major shortcoming of, however, is the computational complexity associated with solving for the MPE. When there are more than a few agents participating in the game and/or more than a few states per agent, the curse of dimensionality renders dynamic programming algorithms intractable.

In this paper we consider a class of stochastic dynamic games where the state of an agent captures its competitive advantage. Our main motivation is to consider an important class of models in applied economics, namely, dynamic industry models of imperfect competition. However, we believe our methods can be useful in other contexts as well. To clarify the type of models we consider, let us describe a specific example of a dynamic industry model. Consider an industry where a group of firms can invest to improve the quality of their products over time. The state of a given firm represents its quality level. The evolution of quality is determined by investment and random shocks. Finally, at every period, given their qualities, firms compete in the product market and receive profits. Many real world industries where, for example, firms invest in R&D or advertising are well described by this model.

In this context, we propose a mean-field approximation approach that dramatically simplifies the computational complexity of stochastic dynamic games. We propose a simple

algorithm for computing an "oblivious" equilibrium in which each agent is assumed to make decisions based only on its own state and knowledge of the long run equilibrium distribution of states, but where agents ignore current information about rivals' states. We prove that, if the distribution of agents obeys a certain "light-tail" condition, when the number of agents becomes large the oblivious equilibrium approximates a MPE. We then derive an error bound that is simple to compute to assess how well the approximation performs for any given number of agents.

We apply our method to analyze dynamic industry models of imperfect competition. We conduct numerical experiments that show that our method works well when there are several hundred firms, and sometimes even tens of firms. Our method, which uses simple code that runs in a couple of minutes on a laptop computer, greatly expands the set of economic problems that can be analyzed computationally.

## 2   A Stochastic Dynamic Game

In this section, we formulate a non-zero sum stochastic dynamic game. The system evolves over discrete time periods and an infinite horizon. We index time periods with nonnegative integers $t \in \mathbb{N}$ ($\mathbb{N} = \{0, 1, 2, \ldots\}$). All random variables are defined on a probability space $(\Omega, \mathcal{F}, \mathcal{P})$ equipped with a filtration $\{\mathcal{F}_t : t \geq 0\}$. We adopt a convention of indexing by $t$ variables that are $\mathcal{F}_t$-measurable.

There are $n$ agents indexed by $S = \{1, ..., n\}$. The state of each agent captures its ability to compete in the environment. At time $t$, the state of agent $i \in S$ is denoted by $x_{it} \in \mathbb{N}$. We define the *system state* $s_t$ to be a vector over individual states that specifies, for each state $x \in \mathbb{N}$, the number of agents at state $x$ in period $t$. We define the state space $\overline{\mathcal{S}} = \left\{ s \in \mathbb{N}^\infty \middle| \sum_{x=0}^\infty s(x) = n \right\}$. For each $i \in S$, we define $s_{-i,t} \in \overline{\mathcal{S}}$ to be the state of the *competitors* of agent $i$; that is, $s_{-i,t}(x) = s_t(x) - 1$ if $x_{it} = x$, and $s_{-i,t}(x) = s_t(x)$, otherwise.

In each period, each agent earns profits. An agent's single period expected profit $\pi_m(x_{it}, s_{-i,t})$ depends on its state $x_{it}$, its competitors' state $s_{-i,t}$ and a parameter $m \in \Re_+$. For example, in the context of an industry model, $m$ could represent the total number of consumers, that is, the size of the pie to be divided among all agents. We assume that for all $x \in \mathbb{N}$, $s \in \mathcal{S}$, $m \in \Re_+$, $\pi_m(x, s) > 0$ and is increasing in $x$. Hence, agents in larger states earn more profits.

In each period, each agent makes a decision. We interpret this decision as an investment to improve the state at the next period. If an agent invests $\mu_{it} \in \Re_+$, then the agent's state at time $t + 1$ is given by, $x_{i,t+1} = x_{it} + w(\mu_{it}, \zeta_{i,t+1})$, where the function $w$ captures the impact of investment on the state and $\zeta_{i,t+1}$ reflects uncertainty in the outcome of investment. For example, in the context of an industry model, uncertainty may arise due to the risk associated with a research endeavor or a marketing campaign. We assume that for all $\zeta$, $w(\mu, \zeta)$ is nondecreasing in $\mu$. Hence, if the amount invested is larger it is more likely the agent will transit next period to a better state. The random variables $\{\zeta_{it} | t \geq 0, i \geq 1\}$ are i.i.d.. We denote the unit cost of investment by $d$.

Each agent aims to maximize expected net present value. The interest rate is assumed to be positive and constant over time, resulting in a constant discount factor of $\beta \in (0, 1)$ per time period. The equilibrium concept we will use builds on the notion of a Markov perfect equilibrium (MPE), in the sense of [3]. We further assume that equilibrium is symmetric, such that all agents use a common stationary strategy. In particular, there is a function $\mu$ such that at each time $t$, each agent $i \in S$ makes a decision $\mu_{it} = \mu(x_{it}, s_{-i,t})$. Let $\mathcal{M}$ denote the set of strategies such that an element $\mu \in \mathcal{M}$ is a function $\mu : \mathbb{N} \times \mathcal{S} \to \Re_+$.

We define the value function $V(x, s|\mu', \mu)$ to be the expected net present value for an agent at state $x$ when its competitors' state is $s$, given that its competitors each follows a common strategy $\mu \in \mathcal{M}$, and the agent itself follows strategy $\mu' \in \mathcal{M}$. In particular,

$$V(x, s|\mu', \mu) = E_{\mu', \mu}\left[\sum_{k=t}^{\infty} \beta^{k-t}\left(\pi(x_{ik}, s_{-i,k}) - d\iota_{ik}\right)\Big|x_{it} = x, s_{-i,t} = s\right],$$

where $i$ is taken to be the index of an agent at state $x$ at time $t$, and the subscripts of the expectation indicate the strategy followed by agent $i$ and the strategy followed by its competitors. In an abuse of notation, we will use the shorthand, $V(x, s|\mu) \equiv V(x, s|\mu, \mu)$, to refer to the expected discounted value of profits when agent $i$ follows the same strategy $\mu$ as its competitors.

An equilibrium to our model comprises a strategy $\mu \in \mathcal{M}$ that satisfy the following condition:

$$(2.1) \qquad \sup_{\mu' \in \mathcal{M}} V(x, s|\mu', \mu) = V(x, s|\mu) \qquad \forall x \in \mathbb{N}, \ \forall s \in \overline{\mathcal{S}}.$$

Under some technical conditions, one can establish existence of an equilibrium in pure strategies [4]. With respect to uniqueness, in general we presume that our model may have multiple equilibria. Dynamic programming algorithms can be used to optimize agent strategies, and equilibria to our model can be computed via their iterative application. However, these algorithms require compute time and memory that grow proportionately with the number of relevant system states, which is often intractable in contexts of practical interest. This difficulty motivates our alternative approach.

## 3    Oblivious Equilibrium

We will propose a method for approximating MPE based on the idea that when there are a large number of agents, simultaneous changes in individual agent states can average out because of a law of large numbers such that the normalized system state remains roughly constant over time. In this setting, each agent can potentially make near-optimal decisions based only on its own state and the long run average system state. With this motivation, we consider restricting agent strategies so that each agent's decisions depend only on the agent's state. We call such restricted strategies *oblivious* since they involve decisions made without full knowledge of the circumstances — in particular, the state of the system. Let $\tilde{\mathcal{M}} \subset \mathcal{M}$ denote the set of oblivious strategies. Since each strategy $\mu \in \tilde{\mathcal{M}}$ generates decisions $\mu(x, s)$ that do not depend on $s$, with some abuse of notation, we will often drop the second argument and write $\mu(x)$.

Let $\tilde{s}_\mu$ be the long-run expected system state when all agents use an oblivious strategy $\mu \in \tilde{\mathcal{M}}$. For an oblivious strategy $\mu \in \tilde{\mathcal{M}}$ we define an *oblivious value function*

$$\tilde{V}(x|\mu', \mu) = E_{\mu'}\left[\sum_{k=t}^{\infty} \beta^{k-t}\left(\pi(x_{ik}, \tilde{s}_\mu) - d\iota_{ik}\right)\Big|x_{it} = x\right].$$

This value function should be interpreted as the expected net present value of an agent that is at state $x$ and follows oblivious strategy $\mu'$, under the assumption that its competitors' state will be $\tilde{s}_\mu$ for all time. Again, we abuse notation by using $\tilde{V}(x|\mu) \equiv \tilde{V}(x|\mu, \mu)$ to refer to the oblivious value function when agent $i$ follows the same strategy $\mu$ as its competitors.

We now define a new solution concept: an *oblivious equilibrium* consists of a strategy $\mu \in \tilde{\mathcal{M}}$ that satisfy the following condition:

$$(3.1) \qquad \sup_{\mu' \in \tilde{\mathcal{M}}} \tilde{V}(x|\mu', \mu) = \tilde{V}(x|\mu), \qquad \forall x \in \mathbb{N}.$$

In an oblivious equilibrium firms optimize an oblivious value function assuming that its competitors' state will be $\tilde{s}_\mu$ for all time. The optimal strategy obtained must be $\mu$. It is straightforward to show that an oblivious equilibrium exists under mild technical conditions. With respect to uniqueness, we have been unable to find multiple oblivious equilibria in any of the applied problems we have considered, but similarly with the case of MPE, we have no reason to believe that in general there is a unique oblivious equilibrium.

## 4 Asymptotic Results

In this section, we establish asymptotic results that provide conditions under which oblivious equilibria offer close approximations to MPE as the number of agents, $n$, grow. We consider a sequence of systems indexed by the one period profit parameter $m$ and we assume that the number of agents in system $m$ is given by $n^{(m)} = am$, for some $a > 0$. Recall that $m$ represents, for example, the total pie to be divided by the agents so it is reasonable to increase $n^{(m)}$ and $m$ at the same rate.

We index functions and random variables associated with system $m$ with a superscript $(m)$. From this point onward we let $\tilde{\mu}^{(m)}$ denote an oblivious equilibrium for system $m$. Let $V^{(m)}$ and $\tilde{V}^{(m)}$ represent the value function and oblivious value function, respectively, when the system is $m$. To further abbreviate notation we denote the expected system state associated with $\tilde{\mu}^{(m)}$ by $\tilde{s}^{(m)} \equiv \tilde{s}_{\tilde{\mu}^{(m)}}$. The random variable $s_t^{(m)}$ denotes the system state at time $t$ when every agent uses strategy $\tilde{\mu}^{(m)}$. We denote the invariant distribution of $\{s_t^{(m)} : t \geq 0\}$ by $q^{(m)}$. In order to simplify our analysis, we assume that the initial system state $s_0^{(m)}$ is sampled from $q^{(m)}$. Hence, $s_t^{(m)}$ is a stationary process; $s_t^{(m)}$ is distributed according to $q^{(m)}$ for all $t \geq 0$. It will be helpful to decompose $s_t^{(m)}$ according to $s_t^{(m)} = f_t^{(m)} n^{(m)}$, where $f_t^{(m)}$ is the random vector that represents the fraction of agents in each state. Similarly, let $\tilde{f}^{(m)} \equiv E[f_t^{(m)}]$ denote the expected fraction of agents in each state. With some abuse of notation, we define $\pi_m(x_{it}, f_{-i,t}, n) \equiv \pi_m(x_{it}, n \cdot f_{-i,t})$. We assume that for all $x \in \mathbb{N}$, $f \in \mathcal{S}_1$, $\pi_m(x, f, n^{(m)}) = \Theta(1)$, where $\mathcal{S}_1 = \{f \in \Re_+^\infty | \sum_{x \in \mathbb{N}} f(x) = 1\}$. If $m$ and $n^{(m)}$ grow at the same rate, one period profits remain positive and bounded.

Our aim is to establish that, under certain conditions, oblivious equilibria well-approximate MPE as $m$ grows. We define the following concept to formalize the sense in which this approximation becomes exact.

**Definition 4.1.** *A sequence $\tilde{\mu}^{(m)} \in \mathcal{M}$ possesses the asymptotic Markov equilibrium (AME) property if for all $x \in \mathbb{N}$,*

$$\lim_{m \to \infty} E_{\tilde{\mu}^{(m)}} \left[ \sup_{\mu' \in \mathcal{M}} V^{(m)}(x, s_t^{(m)} | \mu', \tilde{\mu}^{(m)}) - V^{(m)}(x, s_t^{(m)} | \tilde{\mu}^{(m)}) \right] = 0 \ .$$

The definition of AME assesses approximation error at each agent state $x$ in terms of the amount by which an agent at state $x$ can increase its expected net present value by deviating from the oblivious equilibrium strategy $\tilde{\mu}^{(m)}$, and instead following an optimal (non-oblivious) best response that keeps track of the true system state. The system states are averaged according to the invariant distribution.

It may seem that the AME property is always obtained because $n^{(m)}$ is growing to infinity. However, recall that each agent state reflects its competitive advantage and if there are agents that are too "dominant" this is not necessarily the case. To make this idea more concrete, let us go back to our industry example where firms invest in quality. Even when there are a large number of firms, if the market tends to be concentrated — for example, if the market is usually dominated by a single firm with a an extremely high quality —

the AME property is unlikely to hold. To ensure the AME property, we need to impose a "light-tail" condition that rules out this kind of domination.

Note that $\frac{d \ln \pi_m(y, f, n)}{df(x)}$ is the semi-elasticity of one period profits with respect to the fraction of agents in state $x$. We define the *maximal absolute semi-elasticity function*:

$$g(x) = \max_{m \in \Re_+, y \in \mathbb{N}, f \in \mathcal{S}_1, n \in \mathbb{N}} \left| \frac{d \ln \pi_m(y, f, n)}{df(x)} \right|.$$

For each $x$, $g(x)$ is the maximum rate of relative change of any agent's single-period profit that could result from a small change in the fraction of agents at state $x$. Since larger competitors tend to have greater influence on agent profits, $g(x)$ typically increases with $x$, and can be unbounded.

Finally, we introduce our light-tail condition. For each $m$, let $\tilde{x}^{(m)} \sim \tilde{f}^{(m)}$, that is, $\tilde{x}^{(m)}$ is a random variable with probability mass function $\tilde{f}^{(m)}$. $\tilde{x}^{(m)}$ can be interpreted as the state of an agent that is randomly sampled from among all agents while the system state is distributed according to its invariant distribution.

**Assumption 4.1.** *For all states $x$, $g(x) < \infty$. For all $\epsilon > 0$, there exists a state $z$ such that*

$$E \left[ g(\tilde{x}^{(m)}) \mathbf{1}_{\{\tilde{x}^{(m)} > z\}} \right] \leq \epsilon,$$

*for all $m$.*

Put simply, the light tail condition requires that states where a small change in the fraction of agents has a large impact on the profits of other agents, must have a small probability under the invariant distribution. In the previous example of an industry where firms invest in quality this typically means that very large firms (and hence high concentration) rarely occur under the invariant distribution.

**Theorem 4.1.** *Under Assumption 4.1 and some other regularity conditions[1], the sequence $\tilde{\mu}^{(m)}$ of oblivious equilibria possesses the AME property.*

## 5  Error Bounds

While the asymptotic results from Section 4 provide conditions under which the approximation will work well as the number of agents grows, in practice one would also like to know how the approximation performs for a particular system. For that purpose we derive performance bounds on the approximation error that are simple to compute via simulation and can be used to asses the accuracy of the approximation for a particular problem instance.

We consider a system $m$ and to simplify notation we suppress the index $m$. Consider an oblivious strategy $\tilde{\mu}$. We will quantify approximation error at each agent state $x \in \mathbb{N}$ by $E \left[ \sup_{\mu' \in \mathcal{M}} V(x, s_t | \mu', \tilde{\mu}) - V(x, s_t | \tilde{\mu}) \right]$. The expectation is over the invariant distribution of $s_t$. The next theorem provides a bound on the approximation error. Recall that $\tilde{s}$ is the long run expected state in oblivious equilibrium ($E[s_t]$). Let $a_x(y)$ be the expected discounted sum of an indicator of visits to state $y$ for an agent starting at state $x$ that uses strategy $\tilde{\mu}$.

**Theorem 5.1.** *For any oblivious equilibrium $\tilde{\mu}$ and state $x \in \mathbb{N}$,*

$$(5.1) \qquad E\left[\Delta V\right] \leq \frac{1}{1 - \beta} E[\Delta \pi(s_t)] + \sum_{y \in \mathbb{N}} a_x(y) \left( \pi(y, \tilde{s}) - E\left[\pi(y, s_t)\right] \right),$$

*where* $\Delta V = \sup_{\mu' \in \mathcal{M}} V(x, s_t | \mu', \tilde{\mu}) - V(x, s_t | \tilde{\mu})$ *and* $\Delta \pi(s) = \max_{y \in \mathbb{N}} (\pi(y, s) - \pi(y, \tilde{s}))$.

The error bound can be easily estimated via simulation algorithms. In particular, note that the bound is not a function of the true MPE or even of the optimal non-oblivious best response strategy.

# 6 Application: Industry Dynamics

Many problems in applied economics are dynamic in nature. For example, models involving the entry and exit of firms, collusion among firms, mergers, advertising, investment in R&D or capacity, network effects, durable goods, consumer learning, learning by doing, and transaction or adjustment costs are inherently dynamic. [1] (hereafter EP) introduced an approach to modeling industry dynamics. See [6] for an overview. Computational complexity has been a limiting factor in the use of this modeling approach. In this section we use our method to expand the set of dynamic industries that can be analyzed computationally.

Even though our results apply to more general models where for example firms make exit and entry decisions, here we consider a particular case of an EP model which itself is a particular case of the model introduced in Section 2. We consider a model of a single-good industry with quality differentiation. The agents are firms that can invest to improve the quality of their product over time. In particular $x_{it}$ is the quality level of firm $i$ at time $t$. $\mu_{it}$ represents represents the amount of money invested by firm $i$ at time $t$ to improve its quality. We assume the one period profit function is derived from a logit demand system and where firms compete setting prices. In this case, $m$ represents the market size. See [5] for more details about the model.

## 6.1 Computational Experiments

In this section, we discuss computational results that demonstrate how our approximation method significantly expands the range of relevant EP-type models like the one previously introduced that can be studied computationally.

First, we propose an algorithm to compute oblivious equilibrium [5]. Whether this algorithm is guaranteed to terminate in a finite number of iterations remains an open issue. However, in over 90% of the numerical experiments we present in this section, it converged in less than five minutes (and often much less than this). In the rest, it converged in less than fifteen minutes.

Our first set of results investigate the behavior of the approximation error bound under several different model specifications. A wide range of parameters for our model could reasonably represent different real world industries of interest. In practice the parameters would either be estimated using data from a particular industry or chosen to reflect an industry under study. We begin by investigating a particular set of representative parameter values. See [5] for the specifications.

For each set of parameters, we use the approximation error bound to compute an upper bound on the percentage error in the value function, $\frac{E[\sup_{\mu' \in \mathcal{M}} V(x,s|\mu',\tilde{\mu}) - V(x,s|\tilde{\mu})]}{E[V(x,s|\tilde{\mu})]]}$, where $\tilde{\mu}$ is the OE strategy and the expectations are taken with respect to $s$. We estimate the expectations using simulation. We compute the previously mentioned percentage approximation error bound for different market sizes $m$ and number of firms $n^{(m)}$. As the market size increases, the number of firms increases and the approximation error bound decreases.

In our computational experiments we found that the most important parameter affecting

the approximation error bounds was the degree of vertical product differentiation, which indicates the importance consumers assign to product quality. In Figure 1 we present our results. When the parameter that measures the level of vertical differentiation is low the approximation error bound is less than 0.5% with just 5 firms, while when the parameter is high it is 5% for 5 firms, less than 3% with 40 firms, and less than 1% with 400 firms.

Figure 1: Percentage approximation error bound for fixed number of firms.

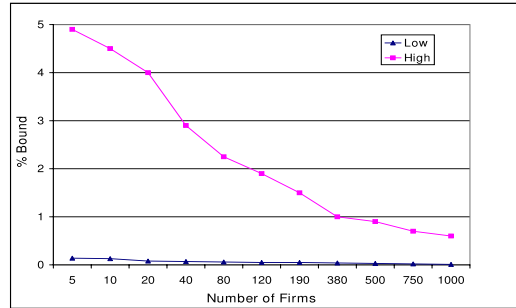

Most economic applications would involve from less than ten to several hundred firms. These results show that the approximation error bound may sometimes be small ($<2\%$) in these cases, though this would depend on the model and parameter values for the industry under study.

Having gained some insight into what features of the model lead to low values of the approximation error bound, the question arises as to what value of the error bounds is required to obtain a good approximation. To shed light on this issue we compare long-run statistics for the same industry primitives under oblivious equilibrium and MPE strategies. A major constraint on this exercise is that it requires the ability to actually compute the MPE, so to keep computation manageable we use four firms here. We compare the average values of several economic statistics of interest under the oblivious equilibrium and the MPE invariant distributions. The quantities compared are: average investment, average producer surplus, average consumer surplus, average share of the largest firm, and average share of the largest two firms. We also computed the actual benefit from deviating and keeping track of the industry state (the actual difference $\frac{E[\sup_{\mu' \in \mathcal{M}} V(x,s|\mu',\tilde{\mu}) - V(x,s|\tilde{\mu})]}{E[V(x,s|\tilde{\mu})]]}$). Note that the the latter quantity should always be smaller than the approximation error bound.

From the computational experiments we conclude the following (see [5] for a table with the results):

1. When the bound is less than 1% the long-run quantities estimated under oblivious equilibrium and MPE strategies are very close.

2. Performance of the approximation depends on the richness of the equilibrium investment process. Industries with a relatively low cost of investment tend to have a symmetric average distribution over quality levels reflecting a rich investment process. In this cases, when the bound is between 1-20%, the long-run quantities estimated under oblivious equilibrium and MPE strategies are still quite close. In industries with high investment cost the industry (system) state tends to be skewed, reflecting low levels of investment. When the bound is above 1% and there is little investment, the long-run quantities can be quite different on a percentage basis (5% to 20%), but still remain fairly close in absolute terms.

3. The performance bound is not tight. For a wide range of parameters the performance bound is as much as 10 to 20 times larger than the actual benefit from deviating.

The previous results suggest that MPE dynamics are well-approximated by oblivious equilibrium strategies when the approximation error bound is small (less than 1-2% and in some cases even up to 20 %). Our results demonstrate that the oblivious equilibrium approximation significantly expands the range of applied problems that can be analyzed computationally.

## 7   Conclusions and Future Research

The goal of this paper has been to increase the set of applied problems that can be addressed using stochastic dynamic games. Due to the curse of dimensionality, the applicability of these models has been severely limited. As an alternative, we proposed a method for approximating MPE behavior using an oblivious equilibrium, where agents make decisions only based on their own state and the long run average system state. We began by showing that the approximation works well asymptotically, where asymptotics were taken in the number of agents. We also introduced a simple algorithm to compute an oblivious equilibrium.

To facilitate using oblivious equilibrium in practice, we derived approximation error bounds that indicate how good the approximation is in any particular problem under study. These approximation error bounds are quite general and thus can be used in a wide class of models. We use our methods to analyze dynamic industry models of imperfect competition and showed that oblivious equilibrium often yields a good approximation of MPE behavior for industries with a couple hundred firms, and sometimes even with just tens of firms.

We have considered very simple strategies that are functions only of an agent's own state and the long run average system state. While our results show that these simple strategies work well in many cases, there remains a set of problems where exact computation is not possible and yet our approximation will not work well either. For such cases, our hope is that our methods will serve as a basis for developing better approximations that use additional information, such as the states of the dominant agents. Solving for equilibria of this type would be more difficult than solving for oblivious equilibria, but is still likely to be computationally feasible. Since showing that such an approach would provide a good approximation is not a simple extension of our results, this will be a subject of future research.

## Footnotes

[1] In particular, we require that the single period profit function is "smooth" as a function of its arguments. See [5] for details.

## References

[1] R. Ericson and A. Pakes. Markov-perfect industry dynamics: A framework for empirical work. *Review of Economic Studies*, 62(1):53 – 82, 1995.

[2] R. L. Goettler, C. A. Parlour, and U. Rajan. Equilibrium in a dynamic limit order market. Forthcoming, Journal of Finance, 2004.

[3] E. Maskin and J. Tirole. A theory of dynamic oligopoly, I and II. *Econometrica*, 56(3):549 – 570, 1988.

[4] U. Doraszelski and M. Satterthwaite. Foundations of Markov-perfect industry dynamics: Existence, purification, and multiplicity. Working Paper, Hoover Institution, 2003.

[5] G. Y. Weintraub, C. L. Benkard, and B. Van Roy. Markov perfect industry dynamics with many firms. Submitted ofr publication, 2005.

[6] A. Pakes. A framework for applied dynamic analysis in i.o. NBER Working Paper 8024, 2000.
